# Applications of multi-resolution neural networks to mammography

**Clay D. Spence and Paul Sajda**
Sarnoff Corporation
CN5300
Princeton, NJ 08543-5300
{cspence, psajda}@sarnoff.com

## Abstract

We have previously presented a coarse-to-fine hierarchical pyramid/neural network (HPNN) architecture which combines multi-scale image processing techniques with neural networks. In this paper we present applications of this general architecture to two problems in mammographic Computer-Aided Diagnosis (CAD). The first application is the detection of microcalcifications. The coarse-to-fine HPNN was designed to learn large-scale context information for detecting small objects like microcalcifications. Receiver operating characteristic (ROC) analysis suggests that the hierarchical architecture improves detection performance of a well established CAD system by roughly 50 %. The second application is to detect mammographic masses directly. Since masses are large, extended objects, the coarse-to-fine HPNN architecture is not suitable for this problem. Instead we construct a fine-to-coarse HPNN architecture which is designed to learn small-scale detail structure associated with the extended objects. Our initial results applying the fine-to-coarse HPNN to mass detection are encouraging, with detection performance improvements of about 36 %. We conclude that the ability of the HPNN architecture to integrate information across scales, both coarse-to-fine and fine-to-coarse, makes it well suited for detecting objects which may have contextual clues or detail structure occurring at scales other than the natural scale of the object.

## 1 Introduction

In a previous paper [8] we presented a coarse-to-fine hierarchical pyramid/neural network (*HPNN*) architecture that combines multi-scale image processing tech-

niques with neural networks to search for small targets in images (see figure 1A). To *search* an image we apply the network at a position and use its output as an estimate of the probability that a target (an object of the class we wish to find) is present there. We then repeat this at each position in the image. In the coarse-to-fine HPNN, the hidden units of networks operating at low resolution or coarse scale learn associated *context* information, since the targets themselves are difficult to detect at low resolution. The context is then passed to networks searching at higher resolution. The use of context can significantly improve detection performance since small objects have few distinguishing features. In the HPNN each of the networks receives information directly from only a small part of several feature images, and so the networks can be relatively simple. The network at the highest resolution integrates the contextual information learned at coarser resolutions to detect the object of interest.

The HPNN architecture can be extended by considering the implications of inverting the information flow in the coarse-to-fine architecture. This fine-to-coarse HPNN would have networks extracting detail structure at fine resolutions of the image and then passing this detail information to networks operating at coarser scales (see figure 1B). For many types of objects, information about the fine structure is important for discriminating between different classes. The fine-to-coarse HPNN is therefore a natural architecture for exploiting fine detail information for detecting extended objects.

In this paper, we present our experiences in applying the HPNN framework to two problems in mammographic Computer-Aided Diagnosis (*CAD*); that of detecting microcalcifications in mammograms and that of detecting malignant masses in mammograms. The coarse-to-fine HPNN architecture is well-suited for the microcalcification problem, while the fine-to-coarse HPNN is suited for mass detection. We evaluate the performance and utility of the HPNN framework by considering its effects on reducing false positive rates in a well characterized CAD system.

The University of Chicago (UofC) has been actively developing mammographic CAD systems for microcalcification and mass detection [6] and has been evaluating their performance clinically. A general block diagram showing the basic processing elements of these CAD systems is shown in figure 2. First, a pre-processing step is used to segment the breast area and increase the overall signal-to-noise levels in the image. Regions of interest (*ROIs*) are defined at this stage, representing local areas of the breast which potentially contain a cluster of microcalcifications or a mass. The next stage typically involves feature extraction and rule-based/heuristic analysis, in order to prune false positives. The remaining ROIs are classified as positive or negative by a statistical classifier or neural network. The CAD system is used as a "second reader", aiding the radiologist by pointing out spots to double check. One of the key requirements of CAD is that false positive rates be low enough that radiologists will not ignore the CAD system output. Therefore it is critical to reduce false positive rates of CAD systems without significant reductions in sensitivity. In this paper we evaluate the HPNN framework within the context of reducing the false positive rates of the UofC CAD systems for microcalcification and mass detection. In both cases the HPNN acts as a post-processor of the UofC CAD system.

## 2   Microcalcification detection

Microcalcifications are calcium deposits in breast tissue that appear as very small bright dots in mammograms. Clusters of microcalcifications frequently occur around tumors. Unfortunately microcalcification clusters are sometimes missed, since they

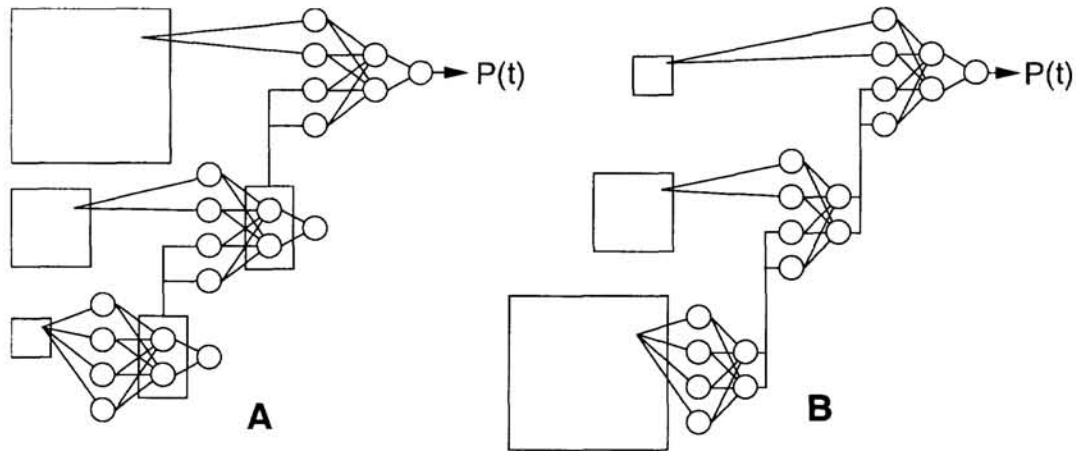

Figure 1: Hierarchical pyramid/neural network architectures for (A) detecting microcalcifications and (B) detecting masses. In (A) context is propagated from low to high resolution via the hidden units of low resolution networks. In (B) small scale detail information is propagated from high to low resolution. In both cases the output of the last integration network is an estimate of the probability that a target is present.

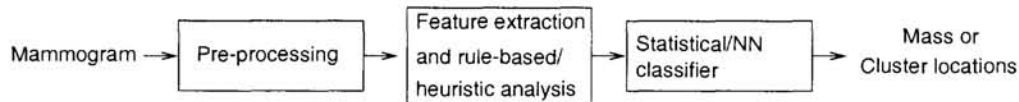

Figure 2: Block diagram for a typical CAD detection system.

can be quite subtle and the radiologists can only spend about a minute evaluating a patient's mammograms.

Data used for the microcalcification experiments was provided by The University of Chicago. The first set of data consists of 50 true positive and 86 false positive ROIs. These ROIs are 99x99 pixels and digitized at 100 micron resolution. A second set of data from the UofC clinical testing database included 47 true positives and 103 false positives, also 99x99 and sampled at 100 micron resolution.

We trained the coarse-to-fine HPNN architecture in figure 1A as a detector for individual calcifications. For each level in the pyramid a network is trained, beginning with the network at low resolution. The network at a particular pyramid level is applied to one pixel at a time in the image at that resolution, and so produces an output at each pixel. All of the networks are trained to detect microcalcifications, however, at low resolutions the microcalcifications are not directly detectable. To achieve better than chance performance, the networks at those levels must learn something about the context in which microcalcifications appear. To integrate context information with the other features the outputs of hidden units from low resolution networks are propagated hierarchically as inputs to networks operating at higher resolutions.

Input to the neural networks come from an integrated feature pyramid ($IFP$) [1]. To construct the IFP, we used steerable filters [3] to compute local orientation energy. The steering properties of these filters enable the direct computation of the orientation having maximum energy. We constructed features which represent, at each pixel location, the maximum energy (energy at $\theta_{max}$), the energy at the

| cc | HPNN | | | | Chicago NN | | | |
|---|---|---|---|---|---|---|---|---|
| | $A_z$ | $\sigma_{A_z}$ | $FPF$ TPF=1.0 | $\sigma_{FPF}$ | $A_z$ | $\sigma_{A_z}$ | $FPF$ TPF=1.0 | $\sigma_{FPF}$ |
| 1 | .93 | .03 | .24 | .11 | .88 | .04 | .50 | .11 |
| 2 | .94 | .02 | .21 | .11 | .91 | .02 | .43 | .10 |
| 3 | .94 | .03 | .39 | .19 | .91 | .03 | .48 | .19 |
| 4 | .93 | .03 | .48 | .15 | .90 | .05 | .56 | .21 |
| 5 | .93 | .03 | .51 | .06 | .88 | .05 | .68 | .21 |

Table 1: Comparison of HPNN and Chicago networks.

orientation perpendicular to $\theta_{max}$ ($\theta_{max} - 90^o$), and the energy at the diagonal (energy at $\theta_{max} - 45^o$).[1] The resulting features are input into the coarse-to-fine network hierarchy.

In examining the truth data for the ROI data set, we found that the experts who specified the microcalcification positions often made errors in these positions of up to $\pm 2$ pixels of the correct position. To take this uncertainty in position into account, we used the following error function

$$E_{UOP} = - \sum_{p \in Pos} \log\left(1 - \prod_{x \in p}(1 - y(x))\right) - \sum_{x \in Neg} \log(1 - y(x)) \qquad (1)$$

which we have called the Uncertain Object Position (*UOP*) error function [7].[2] ($y(x)$ is the network's output when applied to position $x$.) It is essentially the cross-entropy error, but for positive examples the probability of generating a positive output ($y(x)$, in this case) has been replaced by the probability of generating at least one positive output in a region or set of pixels $p$ in the image. In our case each $p$ is a five-by-five pixel square centered on the location specified by the expert. To this we added the standard weight decay regularization term. The regularization constant was adjusted to minimize the ten-fold cross-validation error.

The coarse-to-fine HPNN was applied to each input ROI, and an image was constructed from the output of the Level 0 network at each pixel. Each of these pixel values is the network's estimate of the probability that a microcalcification is present there. Training and testing were done using as jackknife protocol [5], whereby one half of the data (25 TPs and 43 FPs) was used for training and the other half for testing. We used five different random splits of the data into training and test sets. For a given ROI, the probability map produced by the network was thresholded at a given value to produce a binary detection map. Region growing was used to count the number of distinct detected regions. The ROI was classified as a positive if the number of regions was greater than or equal to a certain cluster criterion.

Table 1 compares ROC results for the HPNN and another network that had been used in the University of Chicago CAD system [9] using five different cluster criterion (cc). Reported are the area under the ROC curve ($A_z$), the standard deviation of $A_z$ across the subsets of the jackknife ($\sigma_{A_z}$), the false positive fraction at a true positive fraction of 1.0 ($FPF@TPF = 1.0$) and the standard deviation of the FPF across the subsets of the jackknife ($\sigma_{FPF}$). $A_z$ and $FPF@TPF = 1.0$ represent

the averages of the subsets of the jackknife. Note that both networks operate best when the cluster criterion is set to two. For this case the HPNN has a higher $A_z$ than the Chicago network while also halving the false positive rate. This difference, between the two networks' $A_z$ and $FPF$ values, is statistically significant (z-test; $p_{A_z} = .0018$, $p_{FPF} = .00001$).

A second set of data was also tested. 150 ROIs taken from a clinical prospective study and classified as positive by the full Chicago CAD system (including the Chicago neural network) were used to test the HPNN. Though the Chicago CAD system classified all 150 ROIs as positive, only 47 were in fact positive while 103 were negatives. We applied the HPNN trained on the entire previous data set to this new set of ROIs. The HPNN was able to reclassify 47/103 negatives as negative, without loss in sensitivity (no false negatives were introduced).

On examining the negative examples rejected by the coarse-to-fine HPNN, we found that many of these ROIs contained linear, high-contrast structure which would otherwise be false positives for the Chicago network. The Chicago neural network presumably interprets the "peaks" on the linear structure as calcifications. However because the coarse-to-fine HPNN also integrates information from low resolution it can associate these "peaks" with the low-resolution linear structure and reject them.

## 3   Mass detection

Although microcalcifications are an important cue for malignant masses in mammograms, they are not visible or even present in all cases. Thus mammographic CAD systems include algorithms to directly detect the presence of masses. We have started to apply a fine-to-coarse HPNN architecture to detect malignant masses in digitized mammograms. Radiologists often distinguish malignant from benign masses based on the detailed shape of the mass border and the presence of spicules alone the border. Thus to integrate this high resolution information to detect malignant masses, which are extended objects, we apply the fine-to-coarse HPNN of figure 1B.

As for microcalcifications, we apply the HPNN as a post-processor, but here it processes the output of the mass-detection component of UofC CAD system. The data in our study consists of 72 positive and 100 negative ROIs. These are 256-by-256 pixels and are sampled at 200 micron resolution.

At each level of the fine-to-coarse HPNN several hidden units process the feature images. The outputs of each unit at all of the positions in an image make up a new feature image. This is reduced in resolution by the usual pyramid blur-and-subsample operation to make an input feature image for the network units at the next lower resolution. We trained the entire fine-to-coarse HPNN as one network instead of training a network for each level, one level at a time. This training is quite straightforward. Back-propagating error through the network units is the same as in conventional networks. We must also back-propagate through the pyramid reduction operation, but this is linear and therefore quite simple. In addition we use the same UOP error function (Equation 1) used to train the coarse-to-fine architecture. The rationale for this application of the UOP error function is that the truth data specifies the location of the center of the mass at the highest resolution. However, because of the sub-sampling the center cannot be unambiguously assigned to a particular pixel at low resolution.

The features input to the fine-to-coarse HPNN are filtered versions of the image, with filter kernels given by $\psi_{q,p}(r, \theta) = \left(\frac{q!}{\pi(q+|p|)!}\right)^{1/2} r^{|p|} e^{-r^2/2} L_q^{|p|}(r^2) e^{ip\phi}$ in polar

| Sensitivity | Coarse-to-Fine HPNN Microcalcification | Fine-to-Coarse HPNN Mass |
|---|---|---|
| 100 % | 45 % | 32 % |
| 95 % | 47 % | 36 % |
| 90 % | 63 % | 40 % |
| 80 % | 69 % | 78 % |

Table 2: Detector Specificity (% reduction in false positive rate of UofC CAD system).

coordinates, with $(q,p) \in \{(0,1),(1,0),(0,2)\}$. These are combinations of derivatives of Gaussians, and can be written as combinations of separable filter kernels (products of purely horizontal and vertical filters), so they can be computed at relatively low cost. They are also easy to steer, since this is just multiplication by a complex phase factor. We steered these in the radial and tangential directions relative to the tentative mass centers, and used the real and imaginary parts and their squares and products as features. The center coordinates of the are generated by the earlier stages of the CAD system. These features were extracted at each level of the Gaussian pyramid representation of the mass ROI, and used as inputs only to the network units at the same level.

The fine-to-coarse HPNN is quite similar to the convolution network proposed by Le Cun, et al [2], however with a few notable differences. The fine-to-coarse HPNN receives as inputs preset features extracted from the image (in this case radial and tangential gradients) at each resolution, compared to the convolution network, whose inputs are the original pixel values at the highest resolution. Secondly, in the fine-to-coarse HPNN, the inputs to a hidden unit at a particular position are the pixel values at that position in each of the feature images, one pixel value per feature image. Thus the HPNN's hidden units do not learn linear filters, except as linear combinations of the filters used to form the features. Finally the fine-to-coarse HPNN is trained using the UOP error function, which is not used in the Le Cun network.

Currently our best performing fine-to-coarse HPNN system for mass detection has two hidden units per pyramid level. This gives an ROC area of $A_z = 0.85$ and eliminates 36 % of the false-positives at a cost of missing 5 % of the actual positives. To improve performance further, we are investigating different regularizers, richer feature sets, and more complex architectures, i.e., more hidden units.

## 4   Conclusion

We have presented the application of multi-resolution neural network architectures to two problems in computer-aided diagnosis, the detection of microcalcifications in mammograms and the direct detection of malignant masses in mammograms. A summary of the performance of these architectures is given in Table 2. In the case of microcalcifications, the coarse-to-fine HPNN architecture successfully discovered large-scale context information that improves the system's performance in detecting small objects. A coarse-to-fine HPNN has been directly integrated with the UofC CAD system for microcalcification detection and the complete system is undergoing clinical evaluation.

In the case of malignant masses, a fine-to-coarse HPNN architecture was used to exploit information from fine resolution detail which could be used to differentiate

malignant from benign masses. The results of this network are encouraging, but additional improvement is needed. In general, we have found that the multi-resolution HPNNs are a useful class of network architecture for exploiting and integrating information at multiple scales.

## 5   Acknowledgments

This work was funded by the National Information Display Laboratory, DARPA through ONR contract No. N00014-93-C-0202, and the Murray Foundation. We would like to thank Drs. Robert Nishikawa and Maryellen Giger of The University of Chicago for useful discussions and providing the data.

## Footnotes

[1]We found that the energies in the two diagonal directions were nearly identical.

[2]Keeler et al. [4] developed a network for object recognition that had some similarities to the UOP error. In fact the way in which the outputs of units are combined for their error function can be shown to be an approximation to the UOP error.

## References

[1] Peter Burt. Smart sensing within a pyramid vision machine. *Proceedings IEEE*, 76(8):1006–1015, 1988. Also in *Neuro-Vision Systems*, Gupta and Knopf, eds., 1994.

[2] Y. Le Cun, B. Boser, J. S. Denker, and D. Henderson. Handwritten digit recognition with a back-propagation network. In David S. Touretzky, editor, *Advances in Neural Information Processing Systems 2*, pages 396–404, 2929 Campus Drive, San Mateo, CA 94403, 1991. Morgan-Kaufmann Publishers.

[3] William T. Freeman and Edward H. Adelson. The design and use of steerable filters. *IEEE Transactions on Pattern Analysis and Machine Intelligence*, PAMI-13(9):891–906, 1991.

[4] James D. Keeler, David E. Rumelhart, and Wee-Keng Leow. Integrated segmentation and recognition of hand-printed numerals. In Richard P. Lippmann, John E. Moody, and David S. Touretzky, editors, *Advances in Neural Information Processing Systems 3*, pages 557–563, 2929 Campus Drive, San Mateo, CA 94403, 1991. Morgan-Kaufmann Publishers.

[5] Charles Metz. Current problems in ROC analysis. In *Proceedings of the Chest Imaging Conference*, pages 315–33, Madison, WI, November 1988.

[6] R. M. Nishikawa, R. C. Haldemann, J. Papaioannou, M. L. Giger, P. Lu, R. A. Schmidt, D. E. Wolverton, U. Bick, and K. Doi. Initial experience with a prototype clinical intelligent mammography workstation for computer-aided diagnosis. In Murray H. Loew and Kenneth M. Hanson, editors, *Medical Imaging 1995*, volume 2434, pages 65–71, P.O. Box 10, Bellingham WA 98227-0010, 1995. SPIE.

[7] Clay D. Spence. Supervised learning of detection and classification tasks with uncertain training data. In *Image Understanding Workshop*. ARPA, 1996. This Volume.

[8] Clay D. Spence, John C. Pearson, and Jim Bergen. Coarse-to-fine image search using neural networks. In Gerald Tesauro, David S. Touretzky, and Todd K. Leen, editors, *Advances in Neural Information Processing Systems 7*, pages 981–988, Massachusetts Institute of Technology, Cambridge, MA 02142, 1994. MIT Press.

[9] W. Zhang, K. Doi, M. L. Giger, Y. Wu, R. M. Nishikawa, and R. Schmidt. Computerized detection of clustered microcalcifications in digital mammograms using a shift-invariant artificial neural network. *Medical Physics*, 21(4):517–524, April 1994.